# Modeling Social Annotation Data
# with Content Relevance using a Topic Model

**Tomoharu Iwata**     **Takeshi Yamada**     **Naonori Ueda**
NTT Communication Science Laboratories
2-4 Hikaridai, Seika-cho, Soraku-gun, Kyoto, Japan
{iwata,yamada,ueda}@cslab.kecl.ntt.co.jp

## Abstract

We propose a probabilistic topic model for analyzing and extracting content-related annotations from noisy annotated discrete data such as web pages stored in social bookmarking services. In these services, since users can attach annotations freely, some annotations do not describe the semantics of the content, thus they are noisy, i.e. not content-related. The extraction of content-related annotations can be used as a preprocessing step in machine learning tasks such as text classification and image recognition, or can improve information retrieval performance. The proposed model is a generative model for content and annotations, in which the annotations are assumed to originate either from topics that generated the content or from a general distribution unrelated to the content. We demonstrate the effectiveness of the proposed method by using synthetic data and real social annotation data for text and images.

## 1  Introduction

Recently there has been great interest in social annotations, also called collaborative tagging or folksonomy, created by users freely annotating objects such as web pages [7], photographs [9], blog posts [23], videos [26], music [19], and scientific papers [5]. Delicious [7], which is a social bookmarking service, and Flickr [9], which is an online photo sharing service, are two representative social annotation services, and they have succeeded in collecting huge numbers of annotations. Since users can attach annotations freely in social annotation services, the annotations include those that do not describe the semantics of the content, and are, therefore, not content-related [10]. For example, annotations such as 'nikon' or 'canon' in a social photo service often represent the name of the manufacturer of the camera with which the photographs were taken, or annotations such as '2008' or 'november' indicate when they were taken. Other examples of content-unrelated annotations include those designed to remind the annotator such as 'toread', those identifying qualities such as 'great', and those identifying ownership.

Content-unrelated annotations can often constitute noise if used for training samples in machine learning tasks, such as automatic text classification and image recognition. Although the performance of a classifier can generally be improved by increasing the number of training samples, noisy training samples have a detrimental effect on the classifier. We can improve classifier performance if we can employ huge amounts of social annotation data from which the content-unrelated annotations have been filtered out. Content-unrelated annotations may also constitute noise in information retrieval. For example, a user may wish to retrieve a photograph of a Nikon camera rather than a photograph taken by a Nikon camera.

In this paper, we propose a probabilistic topic model for analyzing and extracting content-related annotations from noisy annotated data. A number of methods for automatic annotation have been proposed [1, 2, 8, 16, 17]. However, they implicitly assume that all annotations are related to content,

Table 1: Notation

| Symbol | Description |
|--------|-------------|
| $D$ | number of documents |
| $W$ | number of unique words |
| $T$ | number of unique annotations |
| $K$ | number of topics |
| $N_d$ | number of words in the $d$th document |
| $M_d$ | number of annotations in the $d$th document |
| $w_{dn}$ | $n$th word in the $d$th document, $w_{dn} \in \{1, \cdots, W\}$ |
| $z_{dn}$ | topic of the $n$th word in the $d$th document, $z_{dn} \in \{1, \cdots, K\}$ |
| $t_{dm}$ | $m$th annotation in the $d$th document, $t_{dm} \in \{1, \cdots, T\}$ |
| $c_{dm}$ | topic of the $m$th annotation in the $d$th document, $c_{dm} \in \{1, \cdots, K\}$ |
| $r_{dm}$ | relevance to the content of the $m$th annotation of the $d$th document, $r_{dm} = 1$ if relevant, $r_{dm} = 0$ otherwise |

and to the best of our knowledge, no attempt has been made to extract content-related annotations automatically. The extraction of content-related annotations can improve performance of machine learning and information retrieval tasks. The proposed model can also be used for the automatic generation of content-related annotations.

The proposed model is a generative model for content and annotations. It first generates content, and then generates the annotations. We assume that each annotation is associated with a latent variable that indicates whether it is related to the content or not, and the annotation originates either from the topics that generated the content or from a content-unrelated general distribution depending on the latent variable. The inference can be achieved based on collapsed Gibbs sampling. Intuitively speaking, this approach considers an annotation to be content-related when it is almost always attached to objects in a specific topic. As regards real social annotation data, the annotations are not explicitly labeled as content related/unrelated. The proposed model is an unsupervised model, and so can extract content-related annotations without content relevance labels.

The proposed method is based on topic models. A topic model is a hierarchical probabilistic model, in which a document is modeled as a mixture of topics, and where a topic is modeled as a probability distribution over words. Topic models are successfully used for a wide variety of applications including information retrieval [3, 13], collaborative filtering [14], and visualization [15] as well as for modeling annotated data [2].

The proposed method is an extension of the correspondence latent Dirichlet allocation (Corr-LDA) [2], which is a generative topic model for contents and annotations. Since Corr-LDA assumes that all annotations are related to the content, it cannot be used for separating content-related annotations from content-unrelated ones. A topic model with a background distribution [4] assumes that words are generated either from a topic-specific distribution or from a corpus-wide background distribution. Although this is a generative model for documents without annotations, the proposed model is related to the model in the sense that data may be generated from a topic-unrelated distribution depending on a latent variable.

In the rest of this paper, we assume that the given data are annotated document data, in which the content of each document is represented by words appearing in the document, and each document has both content-related and content-unrelated annotations. The proposed model is applicable to a wide range of discrete data with annotations. These include annotated image data, where each image is represented with visual words [6], and annotated movie data, where each movie is represented by user ratings.

## 2 Proposed method

Suppose that, we have a set of $D$ documents, and each document consists of a pair of words and annotations $(\boldsymbol{w}_d, \boldsymbol{t}_d)$, where $\boldsymbol{w}_d = \{w_{dn}\}_{n=1}^{N_d}$ is the set of words in a document that represents the content, and $\boldsymbol{t}_d = \{t_{dm}\}_{m=1}^{M_d}$ is the set of assigned annotations, or tags. Our notation is summarized in Table 1.

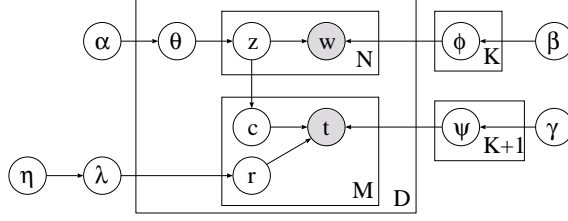

Figure 1: Graphical model representation of the proposed topic model with content relevance.

The proposed topic model first generates the content, and then generates the annotations. The generative process for the content is the same as basic topic models, such as latent Dirichlet allocation (LDA) [3]. Each document has topic proportions $\boldsymbol{\theta}_d$ that are sampled from a Dirichlet distribution. For each of the $N_d$ words in the document, a topic $z_{dn}$ is chosen from the topic proportions, and then word $w_{dn}$ is generated from a topic-specific multinomial distribution $\boldsymbol{\phi}_{z_{dn}}$. In the generative process for annotations, each annotation is assessed as to whether it is related to the content or not. In particular, each annotation is associated with a latent variable $r_{dm}$ with value $r_{dm} = 0$ if annotation $t_{dm}$ is not related to the content; $r_{dm} = 1$ otherwise. If the annotation is not related to the content, $r_{dm} = 0$, annotation $t_{dm}$ is sampled from general topic-unrelated multinomial distribution $\boldsymbol{\psi}_0$. If the annotation is related to the content, $r_{dm} = 1$, annotation $t_{dm}$ is sampled from topic-specific multinomial distribution $\boldsymbol{\psi}_{c_{dm}}$, where $c_{dm}$ is the topic for the annotation. Topic $c_{dm}$ is sampled uniform randomly from topics $\boldsymbol{z}_d = \{z_{dn}\}_{n=1}^{N_d}$ that have previously generated the content. This means that topic $c_{dm}$ is generated from a multinomial distribution, in which $P(c_{dm} = k) = \frac{N_{kd}}{N_d}$, where $N_{kd}$ is the number of words that are assigned to topic $k$ in the $d$th document.

In summary, the proposed model assumes the following generative process for a set of annotated documents $\{(\boldsymbol{w}_d, \boldsymbol{t}_d)\}_{d=1}^D$,

1. Draw relevance probability $\lambda \sim \mathrm{Beta}(\eta)$
2. Draw content-unrelated annotation probability $\boldsymbol{\psi}_0 \sim \mathrm{Dirichlet}(\gamma)$
3. For each topic $k = 1, \cdots, K$:
    (a) Draw word probability $\boldsymbol{\phi}_k \sim \mathrm{Dirichlet}(\beta)$
    (b) Draw annotation probability $\boldsymbol{\psi}_k \sim \mathrm{Dirichlet}(\gamma)$
4. For each document $d = 1, \cdots, D$:
    (a) Draw topic proportions $\boldsymbol{\theta}_d \sim \mathrm{Dirichlet}(\alpha)$
    (b) For each word $n = 1, \cdots, N_d$:
        i. Draw topic $z_{dn} \sim \mathrm{Multinomial}(\boldsymbol{\theta}_d)$
        ii. Draw word $w_{dn} \sim \mathrm{Multinomial}(\boldsymbol{\phi}_{z_{dn}})$
    (c) For each annotation $m = 1, \cdots, M_d$:
        i. Draw topic $c_{dm} \sim \mathrm{Multinomial}(\{\frac{N_{kd}}{N_d}\}_{k=1}^K)$
        ii. Draw relevance $r_{dm} \sim \mathrm{Bernoulli}(\lambda)$
        iii. Draw annotation $t_{dm} \sim \begin{cases} \mathrm{Multinomial}(\boldsymbol{\psi}_0) & \text{if } r_{dm} = 0 \\ \mathrm{Multinomial}(\boldsymbol{\psi}_{c_{dm}}) & \text{otherwise} \end{cases}$

where $\alpha$, $\beta$ and $\gamma$ are Dirichlet distribution parameters, and $\eta$ is a beta distribution parameter. Figure 1 shows a graphical model representation of the proposed model, where shaded and unshaded nodes indicate observed and latent variables, respectively.

As with Corr-LDA, the proposed model first generates the content and then generates the annotations by modeling the conditional distribution of latent topics for annotations given the topics for the content. Therefore, it achieves a comprehensive fit of the joint distribution of content and annotations and finds superior conditional distributions of annotations given content [2].

The joint distribution on words, annotations, topics for words, topics for annotations, and relevance given parameters is described as follows:

$$P(\boldsymbol{W}, \boldsymbol{T}, \boldsymbol{Z}, \boldsymbol{C}, \boldsymbol{R} | \alpha, \beta, \gamma, \eta) = P(\boldsymbol{Z}|\alpha)P(\boldsymbol{W}|\boldsymbol{Z}, \beta)P(\boldsymbol{T}|\boldsymbol{C}, \boldsymbol{R}, \gamma)P(\boldsymbol{R}|\eta)P(\boldsymbol{C}|\boldsymbol{Z}), \quad (1)$$

where $\boldsymbol{W} = \{\boldsymbol{w}_d\}_{d=1}^D$, $\boldsymbol{T} = \{\boldsymbol{t}_d\}_{d=1}^D$, $\boldsymbol{Z} = \{\boldsymbol{z}_d\}_{d=1}^D$, $\boldsymbol{C} = \{\boldsymbol{c}_d\}_{d=1}^D$, $\boldsymbol{c}_d = \{c_{dm}\}_{m=1}^{M_d}$, $\boldsymbol{R} = \{\boldsymbol{r}_d\}_{d=1}^D$, and $\boldsymbol{r}_d = \{r_{dm}\}_{m=1}^{M_d}$. We can integrate out multinomial distribution parameters, $\{\boldsymbol{\theta}_d\}_{d=1}^D$, $\{\boldsymbol{\phi}_k\}_{k=1}^K$ and $\{\boldsymbol{\psi}_{k'}\}_{k'=0}^K$, because we use Dirichlet distributions for their priors, which are conjugate to multinomial distributions. The first term on the right hand side of (1) is calculated by $P(\boldsymbol{Z}|\alpha) = \prod_{d=1}^D \int P(\boldsymbol{z}_d|\boldsymbol{\theta}_d)P(\boldsymbol{\theta}_d|\alpha)d\boldsymbol{\theta}_d$, and we have the following equation by integrating out $\{\boldsymbol{\theta}_d\}_{d=1}^D$,
$P(\boldsymbol{Z}|\alpha) = \left(\frac{\Gamma(\alpha K)}{\Gamma(\alpha)^K}\right)^D \prod_d \frac{\prod_k \Gamma(N_{kd}+\alpha)}{\Gamma(N_d+\alpha K)}$, where $\Gamma(\cdot)$ is the gamma function. Similarly, the second term is given as follows, $P(\boldsymbol{W}|\boldsymbol{Z},\beta) = \left(\frac{\Gamma(\beta W)}{\Gamma(\beta)^W}\right)^K \prod_k \frac{\prod_w \Gamma(N_{kw}+\beta)}{\Gamma(N_k+\beta W)}$, where $N_{kw}$ is the number of times word $w$ has been assigned to topic $k$, and $N_k = \sum_w N_{kw}$. The third term is given as follows, $P(\boldsymbol{T}|\boldsymbol{C},\boldsymbol{R},\gamma) = \left(\frac{\Gamma(\gamma T)}{\Gamma(\gamma)^T}\right)^{K+1} \prod_{k'} \frac{\prod_t \Gamma(M_{k't}+\gamma)}{\Gamma(M_{k'}+\gamma T)}$, where $k' \in \{0, \cdots, K\}$, and $k' = 0$ indicates irrelevant to the content. $M_{k't}$ is the number of times annotation $t$ has been identified as content-unrelated if $k' = 0$, or as content-related topic $k'$ if $k' \neq 0$, and $M_{k'} = \sum_t M_{k't}$. The Bernoulli parameter $\lambda$ can also be integrated out because we use a beta distribution for the prior, which is conjugate to a Bernoulli distribution. The fourth term is given as follows, $P(\boldsymbol{R}|\eta) = \frac{\Gamma(2\eta)}{\Gamma(\eta)^2} \frac{\Gamma(M_0+\eta)\Gamma(M-M_0+\eta)}{\Gamma(M+2\eta)}$, where $M$ is the number of annotations, and $M_0$ is the number of content-unrelated annotations. The fifth term is given as follows, $P(\boldsymbol{C}|\boldsymbol{Z}) = \prod_d \prod_k \left(\frac{N_{kd}}{N_d}\right)^{M'_{kd}}$, where $M'_{kd}$ is the number of annotations that are assigned to topic $k$ in the $d$th document.

The inference of the latent topics $\boldsymbol{Z}$ given content $\boldsymbol{W}$ and annotations $\boldsymbol{T}$ can be efficiently computed using collapsed Gibbs sampling [11]. Given the current state of all but one variable, $z_j$, where $j = (d, n)$, the assignment of a latent topic to the $n$th word in the $d$th document is sampled from,

$$P(z_j = k|\boldsymbol{W},\boldsymbol{T},\boldsymbol{Z}_{\setminus j},\boldsymbol{C},\boldsymbol{R}) \propto \frac{N_{kd\setminus j}+\alpha}{N_{d\setminus j}+\alpha K} \frac{N_{kw_j\setminus j}+\beta}{N_{k\setminus j}+\beta W} \left(\frac{N_{kd\setminus j}+1}{N_{kd\setminus j}} \frac{N_d-1}{N_d}\right)^{M'_{kd}},$$

where $\setminus j$ represents the count when excluding the $n$th word in the $d$th document. Given the current state of all but one variable, $r_i$, where $i = (d, m)$, the assignment of either relevant or irrelevant to the $m$th annotation in the $d$th document is estimated as follows,

$$P(r_i = 0|\boldsymbol{W},\boldsymbol{T},\boldsymbol{Z},\boldsymbol{C},\boldsymbol{R}_{\setminus i}) \propto \frac{M_{0\setminus i}+\eta}{M_{\setminus i}+2\eta} \frac{M_{0t_i\setminus i}+\gamma}{M_{0\setminus i}+\gamma T},$$

$$P(r_i = 1|\boldsymbol{W},\boldsymbol{T},\boldsymbol{Z},\boldsymbol{C},\boldsymbol{R}_{\setminus i}) \propto \frac{M_{\setminus i}-M_{0\setminus i}+\eta}{M_{\setminus i}+2\eta} \frac{M_{c_it_i\setminus i}+\gamma}{M_{c_i\setminus i}+\gamma T}. \tag{2}$$

The assignment of a topic to a content-unrelated annotation is estimated as follows,

$$P(c_i = k|r_i = 0,\boldsymbol{W},\boldsymbol{T},\boldsymbol{Z},\boldsymbol{C}_{\setminus i},\boldsymbol{R}_{\setminus i}) \propto \frac{N_{kd}}{N_d}, \tag{3}$$

and the assignment of a topic to a content-related annotation is estimated as follows,

$$P(c_i = k|r_i = 1,\boldsymbol{W},\boldsymbol{T},\boldsymbol{Z},\boldsymbol{C}_{\setminus i},\boldsymbol{R}_{\setminus i}) \propto \frac{M_{kt_i\setminus i}+\gamma}{M_{k\setminus i}+\gamma T} \frac{N_{kd}}{N_d}. \tag{4}$$

The parameters $\alpha$, $\beta$, $\gamma$, and $\eta$ can be estimated by maximizing the joint distribution (1) by the fixed-point iteration method described in [21].

## 3 Experiments

### 3.1 Synthetic content-unrelated annotations

We evaluated the proposed method quantitatively by using labeled text data from the 20 Newsgroups corpus [18] and adding synthetic content-unrelated annotations. The corpus contains about 20,000 articles categorized into 20 discussion groups. We considered these 20 categories as content-related annotations, and we also randomly attached dummy categories to training samples as content-unrelated annotations. We created two types of training data, 20News1 and 20News2, where the

former was used for evaluating the proposed method when analyzing data with different numbers of content-unrelated annotations per document, and the latter was used with different numbers of unique content-unrelated annotations. Specifically, in the 20News1 data, the unique number of content-unrelated annotations was set at ten, and the number of content-unrelated annotations per document was set at $\{1, \cdots, 10\}$. In the 20News2 data, the unique number of content-unrelated annotations was set at $\{1, \cdots, 10\}$, and the number of content-unrelated annotations per document was set at one. We omitted stop-words and words that occurred only once. The vocabulary size was 52,647. We sampled 100 documents from each of the 20 categories, for a total of 2,000 documents. We used 10 % of the samples as test data.

We compared the proposed method with MaxEnt and Corr-LDA. MaxEnt represents a maximum entropy model [22] that estimates the probability distribution that maximizes entropy under the constraints imposed by the given data. MaxEnt is a discriminative classifier and achieves high performance as regards text classification. In MaxEnt, the hyper-parameter that maximizes the performance was chosen from $\{10^{-3}, 10^{-2}, 10^{-1}, 1\}$, and the input word count vector was normalized so that the sum of the elements was one. Corr-LDA [2] is a topic model for words and annotations that does not take the relevance to content into consideration. For the proposed method and Corr-LDA, we set the number of latent topics, $K$, to 20, and estimated latent topics and parameters by using collapsed Gibbs sampling and the fixed-point iteration method, respectively.

We evaluated the predictive performance of each method using the perplexity of held-out content-related annotations given the content. A lower perplexity represents higher predictive performance. In the proposed method, we calculated the probability of content-related annotation $t$ in the $d$th document given the training samples as follows, $P(t|d, \mathcal{D}) \approx \sum_k \hat{\theta}_{dk} \hat{\psi}_{kt}$, where $\hat{\theta}_{dk} = \frac{N_{kd}}{N_d}$ is a point estimate of the topic proportions for annotations, and $\hat{\psi}_{kt} = \frac{M_{kt} + \gamma}{M_k + \gamma T}$ is a point estimate of the annotation multinomial distribution. Note that no content-unrelated annotations were attached to the test samples. The average perplexities and standard deviations over ten experiments on the 20News1 and 20News2 data are shown in Figure 2 (a). In all cases, when content-unrelated annotations were included, the proposed method achieved the lowest perplexity, indicating that it can appropriately predict content-related annotations. Although the perplexity achieved by MaxEnt was slightly lower than that of the proposed method without content-unrelated annotations, the performance of MaxEnt deteriorated greatly when even one content-unrelated annotation was attached. Since MaxEnt is a supervised classifier, it considers all attached annotations to be content-related even if they are not. Therefore, its perplexity is significantly high when there are fewer content-related annotations per document than unrelated annotations as with the 20News1 data. In contrast, since the proposed method considers the relevance to the content for each annotation, it always offered low perplexity even if the number of content-unrelated annotations was increased. The perplexity achieved by Corr-LDA was high because it does not consider the relevance to the content as in MaxEnt.

We evaluated the performance in terms of extracting content-related annotations. We considered extraction as a binary classification problem, in which each annotation is classified as either content-related or content-unrelated. As the evaluation measurement, we used F-measure, which is the harmonic mean of precision and recall. We compared the proposed method to a baseline method in which the annotations are considered to be content-related if any of the words in the annotations appear in the document. In particular, when the category name is 'comp.graphics', if 'computer' or 'graphics' appears in the document, it is considered to be content-related. We assume that the baseline method knows that content-unrelated annotations do not appear in any document. Therefore, the precision of the baseline method is always one, because the number of false positive samples is zero. Note that this baseline method does not support image data, because words in the annotations never appear in the content. F-measures for the 20News1 and 20News2 data are shown in Figure 2 (b). A higher F-measure represents higher classification performance. The proposed method achieved high F-measures with a wide range of ratios of content-unrelated annotations. All of the F-measures achieved by the proposed method exceeded 0.89, and the F-measure without unrelated annotations was one. This result implies that it can flexibly handle cases with different ratios of content-unrelated annotations. The F-measures achieved by the baseline method were low because annotations might be related to the content even if the annotations did not appear in the document. On the other hand, the proposed method considers that annotations are related to the content when the topic, or latent semantics, of the content and the topic of the annotations are similar even if the annotations did not appear in the document.

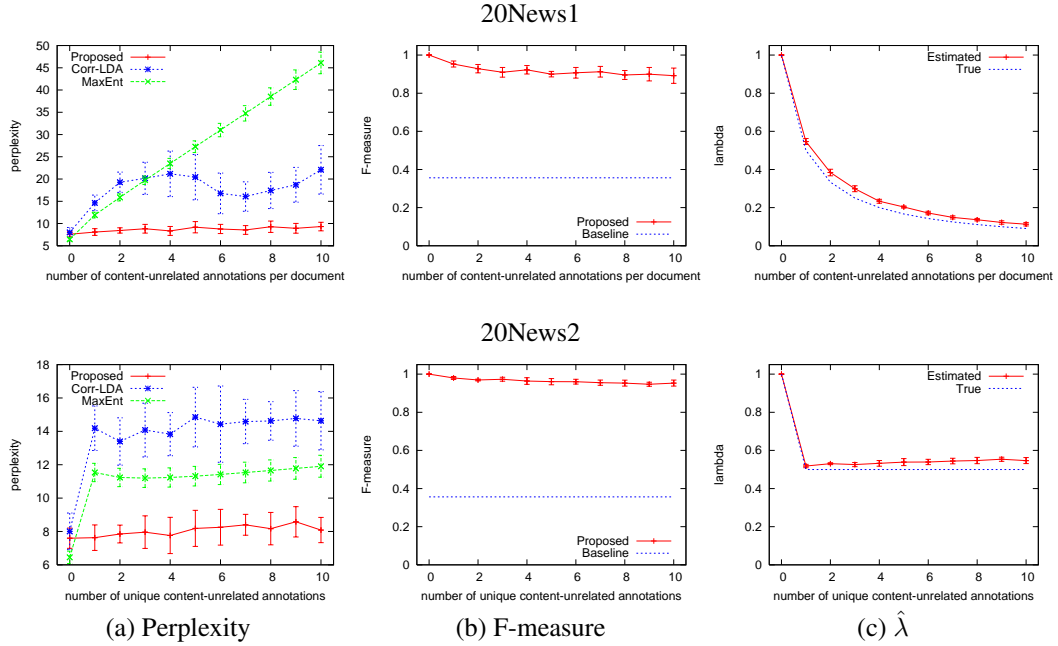

Figure 2: (a) Perplexities of the held-out content-related annotations, (b) F-measures of content relevance, and (c) Estimated content-related annotation ratios in 20News data.

Figure 2 (c) shows the content-related annotation ratios as estimated by the following equation, $\hat{\lambda} = \frac{M - M_0 + \eta}{M + 2\eta}$, with the proposed method. The estimated ratios are about the same as the true ratios.

## 3.2 Social annotations

We analyzed the following three sets of real social annotation data taken from two social bookmarking services and a photo sharing service, namely Hatena, Delicious, and Flickr.

From the Hatena data, we used web pages and their annotations in Hatena::Bookmark [12], which is a social bookmarking service in Japan, that were collected using a similar method to that used in [25, 27]. Specifically, first, we obtained a list of URLs of popular bookmarks for October 2008. We then obtained a list of users who had bookmarked the URLs in the list. Next, we obtained a new list of URLs that had been bookmarked by the users. By iterating the above process, we collected a set of web pages and their annotations. We omitted stop-words and words and annotations that occurred in fewer than ten documents. We omitted documents with fewer than ten unique words and also omitted those without annotations. The numbers of documents, unique words, and unique annotations were 39,132, 8,885, and 43,667, respectively. From the Delicious data, we used web pages and their annotations [7] that were collected using the same method used for the Hatena data. The numbers of documents, unique words, and unique annotations were 65,528, 30,274, and 21,454, respectively. From the Flickr data, we used photographs and their annotations Flickr [9] that were collected in November 2008 using the same method used for the Hatena data. We transformed photo images into visual words by using scale-invariant feature transformation (SIFT) [20] and k-means as described in [6]. We omitted annotations that were attached to fewer than ten images. The numbers of images, unique visual words, and unique annotations were 12,711, 200, and 2,197, respectively. For the experiments, we used 5,000 documents that were randomly sampled from each data set.

Figure 3 (a)(b)(c) shows the average perplexities over ten experiments and their standard deviation for held-out annotations in the three real social annotation data sets with different numbers of topics. Figure 3 (d) shows the result with the Patent data as an example of data without content unrelated annotations. The Patent data consist of patents published in Japan from January to March in 2004, to which International Patent Classification (IPC) codes were attached by experts according to their content. The numbers of documents, unique words, and unique annotations (IPC codes) were 9,557,

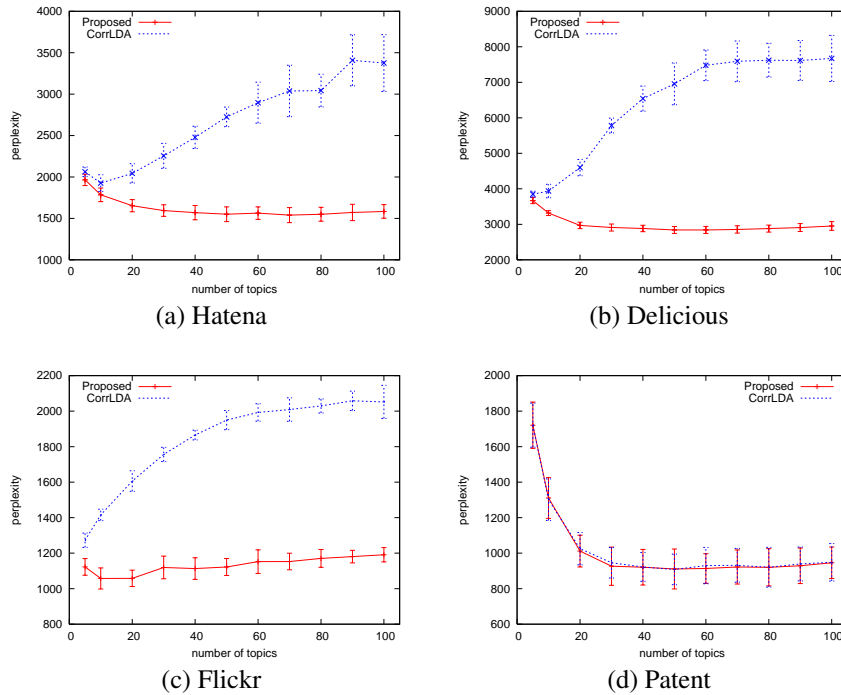

(a) Hatena

(b) Delicious

(c) Flickr

(d) Patent

Figure 3: Perplexities of held-out annotations with different numbers of topics in social annotation data (a)(b)(c), and in data without content unrelated annotations (d).

canada banking toread
London river london history reference imported England
blog ruby rails cell person misc Ruby plugin cpu ajax javascript exif php
future distribution internet prediction Internet computer computers no_tag bandwidth
film Art good mindfuck movies list blog
ricette cucina cooking italy search recipes italian food cook news reference searchengine list italiano links
ruby git diff useful triage imported BookmarksBar blog
SSD toread ssd
c# interview programming C# .net todo language tips microsoft
google gmail googlecalendar Web-2.0 Gmail via:mento.info

Figure 4: Examples of content-related annotations in the Delicious data extracted by the proposed method. Each row shows annotations attached to a document; content-unrelated annotations are shaded.

104,621, and 6,117, respectively. With the Patent data, the perplexities of the proposed method and Corr-LDA were almost the same. On the other hand, with the real social annotation data, the proposed method achieved much lower perplexities than Corr-LDA. This result implies that it is important to consider relevance to the content when analyzing noisy social annotation data. The perplexity of Corr-LDA with social annotation data gets worse as the number of topics increases because Corr-LDA overfits noisy content-unrelated annotations.

The upper half of each table in Table 2 shows probable content-unrelated annotations in the leftmost column, and probable annotations for some topics, which were estimated with the proposed method using 50 topics. The lower half in (a) and (b) shows probable words in the content for each topic. With the Hatena data, we translated Japanese words into English, and we omitted words that had the same translated meaning in a topic. For content-unrelated annotations, words that seemed to be irrelevant to the content were extracted, such as 'toread', 'later', '*', '?', 'imported', '2008', 'nikon', and 'cannon'. Each topic has characteristic annotations and words, for example, Topic1 in the Hatena data is about programming, Topic2 is about games, and Topic3 is about economics. Figure 4 shows some examples of the extraction of content-related annotations.

Table 2: The ten most probable content-unrelated annotations (leftmost column), and the ten most probable annotations for some topics (other columns), estimated with the proposed method using 50 topics. Each column represents one topic. The lower half in (a) and (b) shows probable words in the content.

### (a) Hatena

| unrelated | Topic1 | Topic2 | Topic3 | Topic4 | Topic5 | Topic6 | Topic7 | Topic8 | Topic9 |
|---|---|---|---|---|---|---|---|---|---|
| toread | programming | game | economics | science | food | linux | politics | pc | medical |
| web | development | animation | finance | research | cooking | tips | international | apple | health |
| later | dev | movie | society | biology | gourmet | windows | oversea | iphone | lie |
| great | webdev | Nintendo | business | study | recipe | security | society | hardware | government |
| document | php | movie | economy | psychology | cook | server | history | gadget | agriculture |
| troll | java | event | reading | mathematics | life | network | china | mac | food |
| * | software | xbox360 | investment | pseudoscience | fooditem | unix | world | cupidity | mentalhealth |
| ? | ruby | DS | japan | knowledge | foods | mysql | international | technology | mental |
| summary | opensource | PS3 | money | education | alcohol | mail | usa | ipod | environment |
| memo | softwaredev | animation | company | math | foodie | Apache | news | electronics | science |
| | development | game | year | science | eat | in | japan | yen | rice |
| | web | animation | article | researcher | use | setting | country | product | banana |
| | series | movie | finance | answer | omission | file | usa | digital | medical |
| | hp | story | economics | spirit | water | server | china | pc | diet |
| | technology | work | investment | question | decision | case | politics | support | hospital |
| | management | create | company | human | broil | mail | aso | in | poison |
| | source | PG | day | ehara | face | address | mr | note | eat |
| | usage | mr | management | proof | input | connection | korea | price | incident |
| | project | interesting | information | mind | miss | access | human | equipment | korea |
| | system | world | nikkei | brain | food | security | people | model | jelly |

### (b) Delicious

| reference | money | video | opensource | food | windows | art | shopping | iphone | education |
|---|---|---|---|---|---|---|---|---|---|
| web | finance | music | software | recipes | linux | photo | shop | mobile | learning |
| imported | economics | videos | programming | recipe | sysadmin | photography | Shopping | hardware | books |
| design | business | fun | development | cooking | Windows | photos | home | games | book |
| internet | economy | entertainment | linux | Food | security | Photography | wishlist | iPhone | language |
| online | Finance | funny | tools | Recipes | computer | Art | buy | apple | library |
| cool | financial | movies | rails | baking | microsoft | inspiration | store | tech | school |
| toread | investing | media | ruby | health | network | music | fashion | gaming | teaching |
| tools | bailout | Video | webdev | vegetarian | Linux | foto | gifts | mac | Education |
| blog | finances | film | rubyonrails | diy | ubuntu | fotografia | house | game | research |
| | money | music | project | recipe | windows | art | buy | iphone | book |
| | financial | video | code | food | system | photography | online | apple | legal |
| | credit | link | server | recipes | microsoft | photos | price | ipod | theory |
| | market | tv | ruby | make | linux | camera | cheap | mobile | books |
| | economic | movie | rails | wine | software | vol | product | game | law |
| | october | itunes | source | made | file | digital | order | games | university |
| | economy | film | file | add | server | images | free | pc | students |
| | banks | amazon | version | love | user | 2008 | products | phone | learning |
| | government | play | files | eat | files | photo | rating | mac | education |
| | bank | interview | development | good | ubuntu | tracks | card | touch | language |

### (c) Flickr

| 2008 | dance | sea | autumn | rock | beach | family | island |
|---|---|---|---|---|---|---|---|
| nikon | bar | sunset | trees | house | travel | portrait | asia |
| canon | dc | sky | tree | party | vacation | cute | landscape |
| white | digital | clouds | mountain | park | camping | baby | rock |
| yellow | concert | mountains | fall | inn | landscape | boy | blue |
| red | bands | ocean | garden | coach | texas | kids | tour |
| photo | music | panorama | bortescristian | creature | lake | brown | plant |
| italy | washingtondc | south | geotagged | halloween | cameraphone | closeup | tourguidesoma |
| california | dancing | ireland | mud | mallory | md | 08 | koh |
| color | work | oregon | natura | night | sun | galveston | samui |

## 4 Conclusion

We have proposed a topic model for extracting content-related annotations from noisy annotated data. We have confirmed experimentally that the proposed method can extract content-related annotations appropriately, and can be used for analyzing social annotation data. In future work, we will determine the number of topics automatically by extending the proposed model to a nonparametric Bayesian model such as the Dirichlet process mixture model [24]. Since the proposed method is, theoretically, applicable to various kinds of annotation data, we will confirm this in additional experiments.

# References

[1] K. Barnard, P. Duygulu, D. Forsyth, N. de Freitas, D. M. Blei, and M. I. Jordan. Matching words and pictures. *Journal of Machine Learning Research*, 3:1107–1135, 2003.

[2] D. M. Blei and M. I. Jordan. Modeling annotated data. In *SIGIR '03: Proceedings of the 26th Annual International ACM SIGIR Conference on Research and Development in Information Retrieval*, pages 127–134, 2003.

[3] D. M. Blei, A. Y. Ng, and M. I. Jordan. Latent Dirichlet allocation. *Journal of Machine Learning Research*, 3:993–1022, 2003.

[4] C. Chemudugunta, P. Smyth, and M. Steyvers. Modeling general and specific aspects of documents with a probabilistic topic model. In B. Schölkopf, J. Platt, and T. Hoffman, editors, *Advances in Neural Information Processing Systems 19*, pages 241–248. MIT Press, 2007.

[5] CiteULike. http://www.citeulike.org.

[6] G. Csurka, C. Dance, J. Willamowski, L. Fan, and C. Bray. Visual categorization with bags of keypoints. In *ECCV International Workshop on Statistical Learning in Computer Vision*, 2004.

[7] Delicious. http://delicious.com.

[8] S. Feng, R. Manmatha, and V. Lavrenko. Multiple Bernoulli relevance models for image and video annotation. In *CVPR '04: Proceedings of the IEEE Computer Society Conference on Computer Vision and Pattern Recognition*, volume 2, pages 1002–1009, 2004.

[9] Flickr. http://flickr.com.

[10] S. Golder and B. A. Huberman. Usage patterns of collaborative tagging systems. *Journal of Information Science*, 32(2):198–208, 2006.

[11] T. L. Griffiths and M. Steyvers. Finding scientific topics. *Proceedings of the National Academy of Sciences*, 101 Suppl 1:5228–5235, 2004.

[12] Hatena::Bookmark. http://b.hatena.ne.jp.

[13] T. Hofmann. Probabilistic latent semantic analysis. In *UAI '99: Proceedings of 15th Conference on Uncertainty in Artificial Intelligence*, pages 289–296, 1999.

[14] T. Hofmann. Collaborative filtering via Gaussian probabilistic latent semantic analysis. In *Proceedings of the 26th Annual International ACM SIGIR Conference on Research and Development in Information Retrieval*, pages 259–266. ACM Press, 2003.

[15] T. Iwata, T. Yamada, and N. Ueda. Probabilistic latent semantic visualization: topic model for visualizing documents. In *KDD '08: Proceeding of the 14th ACM SIGKDD International Conference on Knowledge Discovery and Data Mining*, pages 363–371. ACM, 2008.

[16] J. Jeon, V. Lavrenko, and R. Manmatha. Automatic image annotation and retrieval using cross-media relevance models. In *SIGIR '03: Proceedings of the 26th Annual International ACM SIGIR Conference on Research and Development in Information Retrieval*, pages 119–126. ACM, 2003.

[17] J. Jeon and R. Manmatha. Using maximum entropy for automatic image annotation. In *CIVR '04: Proceedings of the 3rd International Conference on Image and Video Retrieval*, pages 24–32, 2004.

[18] K. Lang. NewsWeeder: learning to filter netnews. In *ICML '95: Proceedings of the 12th International Conference on Machine Learning*, pages 331–339, 1995.

[19] Last.fm. http://www.last.fm.

[20] D. G. Lowe. Distinctive image features from scale-invariant keypoints. *International Journal of Computer Vision*, 60(2):91–110, 2004.

[21] T. Minka. Estimating a Dirichlet distribution. Technical report, M.I.T., 2000.

[22] K. Nigam, J. Lafferty, and A. McCallum. Using maximum entropy for text classification. In *Proceedings of IJCAI-99 Workshop on Machine Learning for Information Filtering*, pages 61–67, 1999.

[23] Technorati. http://technorati.com.

[24] Y. W. Teh, M. I. Jordan, M. J. Beal, and D. M. Blei. Hierarchical Dirichlet processes. *Journal of the American Statistical Association*, 101(476):1566–1581, 2006.

[25] X. Wu, L. Zhang, and Y. Yu. Exploring social annotations for the semantic web. In *WWW '06: Proceedings of the 15th International Conference on World Wide Web*, pages 417–426. ACM, 2006.

[26] YouTube. http://www.youtube.com.

[27] D. Zhou, J. Bian, S. Zheng, H. Zha, and C. L. Giles. Exploring social annotations for information retrieval. In *WWW '08: Proceeding of the 17th International Conference on World Wide Web*, pages 715–724. ACM, 2008.

